# Noise suppression based on neurophysiologically-motivated SNR estimation for robust speech recognition

**Jürgen Tchorz**
Medical Physics Group
Oldenburg University
26111 Oldenburg
Germany
*tch@medi.physik.uni-oldenburg.de*

**Michael Kleinschmidt**
Medical Physics Group
Oldenburg University
26111 Oldenburg
Germany

**Birger Kollmeier**
Medical Physics Group
Oldenburg University
26111 Oldenburg
Germany

## Abstract

A novel noise suppression scheme for speech signals is proposed which is based on a neurophysiologically-motivated estimation of the local signal-to-noise ratio (SNR) in different frequency channels. For SNR-estimation, the input signal is transformed into so-called Amplitude Modulation Spectrograms (AMS), which represent both spectral and temporal characteristics of the respective analysis frame, and which imitate the representation of modulation frequencies in higher stages of the mammalian auditory system. A neural network is used to analyse AMS patterns generated from noisy speech and estimates the local SNR. Noise suppression is achieved by attenuating frequency channels according to their SNR. The noise suppression algorithm is evaluated in speaker-independent digit recognition experiments and compared to noise suppression by Spectral Subtraction.

## 1 Introduction

One of the major problems in automatic speech recognition (ASR) systems is their lack of robustness in noise, which severely degrades their usefulness in many practical applications. Several proposals have been made to increase the robustness of ASR systems, e.g. by model compensation or more noise-robust feature extraction [1, 2]. Another method to increase robustness of ASR systems is to suppress the background noise before feature extraction. Classical approaches for single-channel noise suppression are Spectral Subtraction [3] and related schemes, e.g. [4], where

the noise spectrum is usually measured in detected speech pauses and subtracted from the signal. In these approaches, stationarity of the noise has to be assumed while speech is active. Furthermore, portions detected as speech pauses must not contain any speech in order to allow for correct noise measurement. At the same time, all actual speech pauses should be detected for a fast update of the noise measurement. In reality, however, these partially conflicting requirements are often not met.

The noise suppression algorithm outlined in this work directly estimates the local SNR in a range of frequency channels even if speech and noise are present at the same time, i.e., no explicit detection of speech pauses and no assumptions on noise stationarity during speech activity are necessary. For SNR estimation, the input signal is transformed into spectro-temporal input features, which are neurophysiologically-motivated: experiments on amplitude modulation processing in higher stages of the auditory system in mammals show that modulations are represented in "periodotopical" gradients, which are almost orthogonal to the tonotopical organization of center frequencies [5]. Thus, both spectral and temporal information is represented in two-dimensional maps. These findings were applied to signal processing in a binaural noise suppression system [6] with the introduction of so-called Amplitude Modulation Spectrograms (AMS), which contain information on both center frequencies and modulation frequencies. In the present study, the different representations of speech and noise in AMS patterns are detected by a neural network, which estimates the local SNR in each frequency channel. For noise suppression, the frequency bands are attenuated according to the estimated local SNR in the different frequency channels.

The proposed noise suppression scheme is evaluated in isolated-digit recognition experiments. As recognizer, a combination of an auditory-based front end [2] and a locally-recurrent neural network [7] is used. This combination was found to allow for more robust isolated-digit recognition rates, compared to a standard recognizer with mel-cepstral features and HMM modeling [8, 9]. Thus, the recognition experiments in this study were conducted with this particular combination to evaluate whether a further increase of robustness can be achieved with additional noise suppression.

## 2  The recognition system

### 2.1  Noise suppression

Figure 1 shows the processing steps which are performed for noise suppression. To generate AMS patterns which are used for SNR estimation, the input signal (16 kHz sampling rate) is short-term level adjusted, i.e., each 32 ms segment which is later transformed into an AMS pattern is scaled to the same root-mean-square value. The level-adjusted signal is then subdivided into overlapping segments of 4.0 ms duration with a progression of 0.25 ms for each new segment. Each segment is multiplied by a Hanning window and padded with zeros to obtain a frame of 128 samples which is transformed with a FFT into a complex spectrum, with a spectral resolution of 125 Hz. The resulting 64 complex samples are considered as a function of time, i.e., as a band pass filtered complex time signal. Their respective envelopes are extracted by squaring. This envelope signal is again segmented into overlapping segments of 128 samples (32ms) with an overlap of 64 samples. Each segment is multiplied with a Hanning window and padded with zeros to obtain a frame of 256 samples. A further FFT is computed and supplies a modulation spectrum in each frequency channel, with a modulation frequency resolution of 15.6 Hz. By an appropriate summation of neighbouring FFT bins the frequency axis is transformed to a Bark scale with 15 channels, with center frequencies from 100-7300 Hz. The modulation

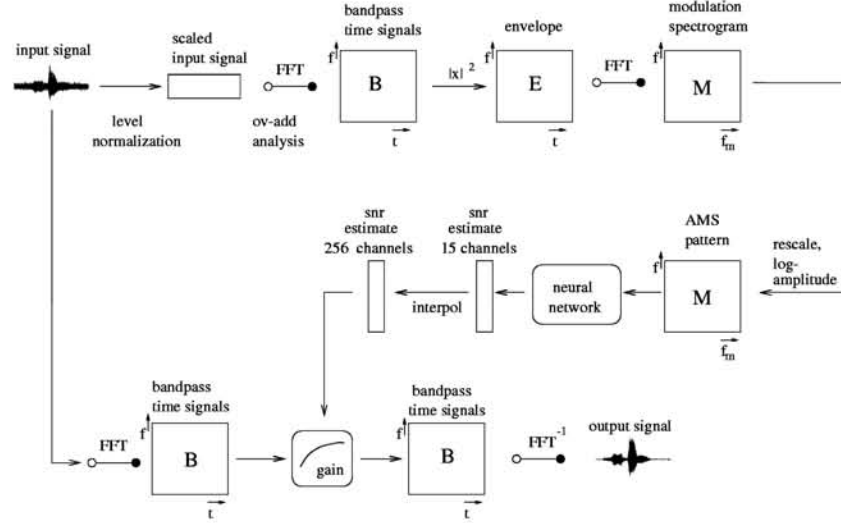

Figure 1: Processing stages of AMS-based noise suppression.

frequency spectrum is scaled logarithmically by appropriate summation, which is motivated by psychoacoustical findings about the shape of auditory modulation filters [10]. The modulation frequency spectrum is restricted to the range between 50-400 Hz and has a resolution of 15 channels. Thus, the fundamental frequency of typical voiced speech is represented in the modulation spectrum. The AMS representation is restricted to a 15 times 15 pattern to limit the amount of training data which is necessary to train the fully connected perceptron. In a last processing step, the amplitude range is log-compressed. Examples for AMS patterns can be seen in Fig. 2. The AMS pattern on the left side was generated from a voiced speech portion. The periodicity at the fundamental frequency (approx. 110 Hz) is represented in each center frequency band. The AMS pattern on the right side was generated from speech simulating noise. The typical spectral tilt can be seen, but there is no structure across modulation frequencies.

For classifying AMS patterns and estimating the narrow-band SNR of each AMS pattern, a feed-forward neural network is employed. The net consists of 225 input neurons (15*15, the AMS resolution of center frequencies and modulation frequencies, respectively), a hidden layer with 160 neurons, and an output layer with 15 neurons. The activity of each output neuron indicates the SNR in one of the 15 center frequency channels. For training, the narrow-band SNRs in 15 channels were measured for each AMS analysis frame of the training material prior to adding speech and noise. The neural network was trained with AMS patterns generated from 72 min of noisy speech from 400 talkers and 41 natural noise types, using the momentum backpropagation algorithm. After training, AMS patterns generated from "unknown" sound material are presented to the network. The 15 output neuron activities that appear for each pattern serve as SNR estimates for the respective frequency channels. In a detailed study on AMS-based broad-band SNR estimation [11] it was shown that harmonicity which is well represented in AMS patterns is the most important cue for the neural network to distinguish between speech and noise. However, harmonicity is not the only cue, as the algorithm allows for reliable discrimination between unvoiced speech and noise. The accuracy of SNR

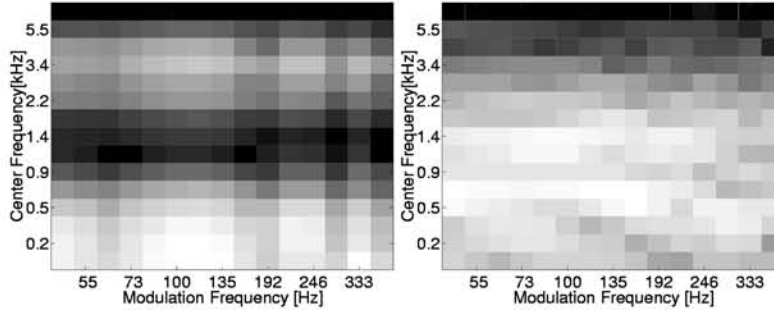

Figure 2: AMS patterns generated from a voiced speech segment (left), and from speech simulating noise (right). Each AMS pattern represents a 32 ms portion of the input signal. Bright and dark areas indicate high and low energies, respectively.

estimation in terms of mean deviation between the actual and the estimated SNR in each frame, for each frequency channel, was determined with "unknown" test data (36 min of noisy speech). The average deviation across all frequency channels was 5.4 dB, with a decrease of accuracy towards higher frequency channels. Sub-band SNR estimates are utilized for noise suppression by attenuating frequency channels according to their local SNR. The gain function which was applied is given by $g_k = (\text{SNR}_k/(\text{SNR}_k + 1))^x$, where $k$ denotes the frequency channel, SNR the signal-to-noise ratio on a linear scale, and $x$ is an exponent which controls the strength of the attenuation, and which was set to 1.5 for the experiments described below.

Noise suppression based on AMS-derived SNR estimation is performed in the FFT-domain. The input signal is segmented into overlapping frames with a window length of 32 ms, and a shift of 16 ms is applied, i.e., each window corresponds to one AMS analysis frame. The FFT is computed in every window. The magnitude in each frequency bin is multiplied by the corresponding gain computed from the AMS-based SNR estimation. The gain in frequency bins which are not covered by the center frequencies from the SNR estimation is linearly interpolated from neighboring estimation frequencies. The phase of the input signal is unchanged and applied to the attenuated magnitude spectrum. An inverse FFT is computed, and the enhanced speech is attained by overlapping and adding.

## 2.2 Auditory-based ASR feature extraction

The front end which is used in the recognition system is based on a quantitative model of the "effective" peripheral auditory processing. The model simulates both spectral and temporal properties of sound processing in the auditory system which were found in psychoacoustical and physiological experiments. The model was originally developed for describing human performance in typical psychoacoustical spectral and temporal masking experiments, e.g., predicting the thresholds in backward, simultaneous, and forward-masking experiments [12, 13]. The main processing stages of the auditory model are gammatone filtering, envelope extraction in each frequency channel, adaptive amplitude compression, and low pass filtering of the envelope in each band. The adaptive compression stage compresses steady-state portions of the input signal logarithmically. Changes like onsets or offsets, in contrast, are transformed linearly. A detailed description of the auditory-based front end is given in [2].

## 2.3 Neural network recognizer

For scoring of the input features, a locally recurrent neural network (LRNN) is employed with three layers of neurons (150 input, 289 hidden, and 10 output neurons). Hidden layer neurons have recurrent connections to their 24 nearest neighbours. The input matrix consists of 5 times the auditory model feature vector with 30 elements, glued together in order to allow the network to memorize a time sequence of input matrices. The network was trained using the Backpropagation-trough-time algorithm with 200 iterations (see [7] for a detailed description of the recognizer).

## 3 Recognition experiments

### 3.1 Setup

The speech material for training of the word models and scoring was taken from the ZIFKOM database of Deutsche Telekom AG. Each German digit was spoken once by 200 different speakers (100 males, 100 females). The recording sessions took place in soundproof booths or quiet offices. The speech material was sampled at 16 kHz.

Three different types of noise were added to the speech material at different signal-to-noise ratios before feature extraction: a) white Gaussian noise, b) speech-simulating noise which is characterized by a long-term speech spectrum and amplitude modulations which reflect an uncorrelated superposition of 6 speakers, and c) background noise recorded in a printing room which strongly fluctuates in both amplitude and spectral shape. The background noises were added to the utterances with signal-to-noise ratios ranging from 20 to -10 dB. The word models were trained with features from 100 undisturbed and unprocessed utterances of each digit. Features for testing were calculated from another 100 utterances of each digit which were distorted by additive noise before preprocessing. The recognition rates were measured without noise suppression and with noise suppression as described in Section 2.1.

For comparison, the recognition rates were measured with noise suppression based on Spectral Subtraction including residual noise reduction [3] before feature extraction. Two methods for noise estimation were applied. In the first method, speech pauses in the noisy signals were detected using Voice Activity Detection (VAD) [14]. The noise measure was updated in speech pauses using a low pass filter with a time constant of 40 ms. In the second method, the noise spectrum was measured in speech pauses which were detected from the *clean* utterances using an energy criterion (thus, perfect speech pause information is provided, which is not available in real applications).

### 3.2 Results

The speaker-independent isolated-digit recognition rates which were obtained in the experiments are plotted in Fig. 3 for three types of background noise as a function of the SNR. In all tested noises, noise suppression with the proposed algorithm increases the recognition rate in comparison with the unprocessed data and with Spectral Subtraction with VAD-based noise measurement. Spectral Subtraction with perfect speech pause detection allows for higher recognition rates than the AMS-based approach in stationary white noise. Here, the noise measure for Spectral Subtraction is very accurate during speech activity and allows for effective noise removal. AMS-based noise suppression estimates the SNR in every analysis frame, and no a priori information on speech-free segments is provided to the algorithm. In

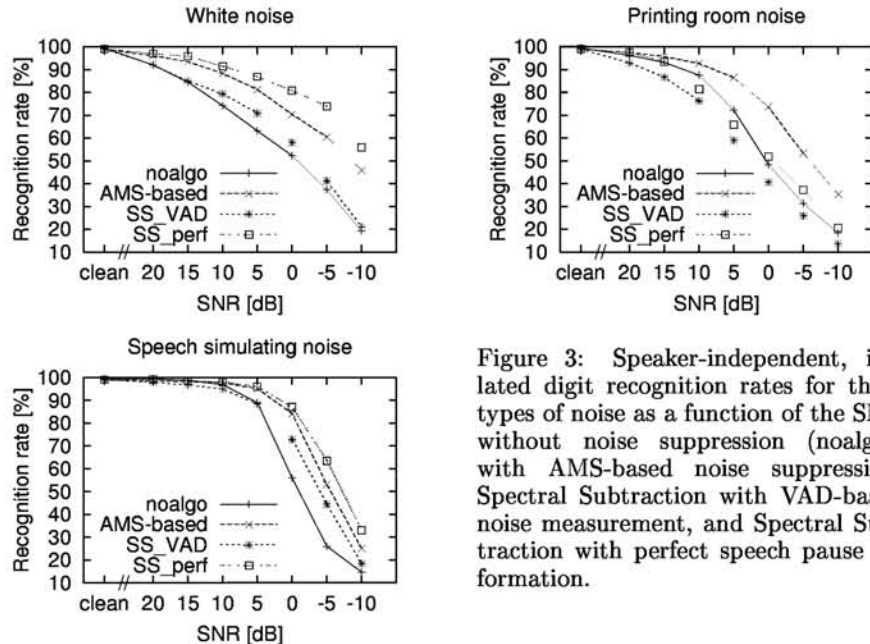

Figure 3: Speaker-independent, isolated digit recognition rates for three types of noise as a function of the SNR without noise suppression (noalgo), with AMS-based noise suppression, Spectral Subtraction with VAD-based noise measurement, and Spectral Subtraction with perfect speech pause information.

speech simulation noise, which fluctuates in level but not in spectral shape, Spectral Subtraction with perfect speech pause detection works slightly better than AMS-based noise suppression. In printing room noise, which fluctuates in both level and spectrum, the AMS-based approach yields the best results. Here, Spectral Subtraction even degrades the recognition rates in some SNRs, compared to the unprocessed data. The noise measure from VAD-based or perfect speech pause detection cannot be updated while speech is active. Thus, an incorrect spectrum is subtracted and leads to artifacts and degraded recognition performance. In clean speech, recognition rates of 99.5% for unprocessed speech, 99.1% after Spectral Subtraction, and 98.9% after AMS-based noise suppression were obtained.

## 4 Discussion

The proposed neurophysiologically-motivated noise suppression scheme was shown to significantly improve digit recognition in noise in comparison with unprocessed data and with Spectral Subtraction using VAD-based noise measures. A perfect speech pause detection (which is not available yet in real systems) allows for a reliable estimation of the noise floor in stationary noise. In non-stationary noise, however, the AMS pattern-based signal classification and noise suppression is advantageous, as it does not depend on speech pause detection and no assumption is necessary about the noise being stationary while speech is active. Spectral Subtraction as described in [3] produces musical tones, i.e. fast fluctuating spectral peaks. The neurophysiologically-based noise suppression scheme outlined in this paper does not produce such fast fluctuating artifacts. In general, a good quality of speech is maintained. The choice of the attenuation exponent $x$ has only little impact on the quality of speech in favourable SNRs. With decreasing SNR, however, there is a tradeoff between the amount of noise suppression and distortions

of the speech. A typical distortion of speech in poor signal-to-noise ratios is an unnatural spectral "coloring", rather than fast fluctuating distortions. In informal tests, most listeners did not have the impression that the algorithm improves speech intelligibility, but clearly preferred the processed signal over the unprocessed one, as the background noise was significantly suppressed without annoying artifacts. Clean speech is almost perfectly preserved after processing. The performance and characteristics of the algorithm of course strongly depends on the training data, as only lttle knowledge on the differences between speech and noise is "hard wired".

## Acknowledgments

We thank Klaus Kasper and Herbert Reininger from Institut für Angewandte Physik, Universität Frankfurt/M. for supplying us with their LRNN implementation.

## References

[1] Hermansky, H. and Morgan, N. (1994). RASTA processing of speech. IEEE Trans. Speech Audio Processing 2(4), pp. 578–589

[2] Tchorz, J. and Kollmeier, B. (1999). A Model of Auditory Perception as Front End for Automatic Speech Recognition. J. Acoust. Soc. Am. 106, pp. 2040–2050

[3] Boll, S. (1979). Suppression of acoustic noise in speech using spectral subtraction. IEEE Trans. Acoust., Speech, Signal Processing 27(2), pp. 113–120

[4] Ephraim, Y. and Malah, M. (1984). Speech enhancement using a minimum mean-square error short-time spectral amplitude estimator. IEEE Trans. Acoust., Speech, Signal Processing 32(6), pp. 1109-1121

[5] Langner, G., Sams, M., Heil, P., and Schulze, H., (1997). Frequency and periodicity are represented in orthogonal maps in the human auditory cortex: evidence from magnetoencephalography. J. Comp. Physiol. A 181, pp. 665–676

[6] Kollmeier, B. and Koch, R., (1994). Speech enhancement based on physiological and psychoacoustical models of modulation perception and binaural interaction. J. Acoust. Soc. Am. 95, pp. 1593–1602

[7] Kasper, K., Reininger, H., Wolf, D., and Wüst, H. (1995). A speech recognizer with low complexity based on RNN. In: Neural Networks for Signal Processing V, Proc. of the IEEE workshop, Cambridge (MA), pp. 272–281

[8] Kasper, K., Reininger, R., and Wolf, D. (1997). Exploiting the potential of auditory preprocessing for robust speech recognition by locally recurrent neural networks. Proc. Int. Conf. Acoustics, Speech and Signal Processing (ICASSP) 2, pp. 1223–1227

[9] Kleinschmidt, M., Tchorz, J., and Kollmeier, B. (2000). Combining speech enhancement and auditory feature extraction for robust speech recognition. Speech Communication, Special issue on robust ASR (accepted)

[10] Ewert, S. and Dau, T. (1999). Frequency selectivity in amplitude-modulation processing. J. Acoust. Soc. Am. (submitted)

[11] Tchorz, J. and Kollmeier, B. (2000). Estimation of the signal-to-noise ratio with amplitude modulation spectrograms. Speech Communication (submitted)

[12] Dau, T., Püschel, D., and Kohlrausch, A. (1996). A quantitative model of the "effective" signal processing in the auditory system: II. Simulations and measurements. J. Acoust. Soc. Am 99, pp. 3623–3631

[13] Dau, T., Kollmeier, B., and Kohlrausch, A. (1997). Modeling auditory processing of amplitude modulation: I. Modulation Detection and masking with narrowband carriers. J. Acoust. Soc. Am 102, pp. 2892–2905

[14] Recommendation ITU-T G.729 Annex B, 1996
